# Reinforcement Learning in Markovian and Non-Markovian Environments

**Jürgen Schmidhuber**
Institut für Informatik
Technische Universität München
Arcistr. 21, 8000 München 2, Germany
schmidhu@tumult.informatik.tu-muenchen.de

## Abstract

This work addresses three problems with reinforcement learning and adaptive neuro-control: 1. Non-Markovian interfaces between learner and environment. 2. On-line learning based on system realization. 3. Vector-valued adaptive critics. An algorithm is described which is based on system realization and on two interacting fully recurrent continually running networks which may learn in parallel. Problems with parallel learning are attacked by 'adaptive randomness'. It is also described how interacting model/controller systems can be combined with *vector-valued* 'adaptive critics' (previous critics have been *scalar*).

## 1 INTRODUCTION

At a given time, an agent with a *non-Markovian interface* to its environment *cannot* derive an optimal next action by considering its current input only. The algorithm described below differs from previous reinforcement algorithms in at least some of the following issues: It has a potential for on-line learning and non-Markovian environments, it is local in time and in principle it allows arbitrary time lags between actions and ulterior consequences; it does not care for something like episode-boundaries, it allows vector-valued reinforcement, it is based on two interacting fully recurrent continually running networks, and it tries to construct a full environmental model - thus providing complete 'credit assignment paths' into the past.

We dedicate one or more conventional input units (called *pain* and *pleasure units*) for the purpose of reporting the actual reinforcement to a fully recurrent control network. Pain and pleasure input units have time-invariant desired values.

We employ the IID-Algorithm (Robinson and Fallside, 1987) for training a fully recurrent model network to model the relationships between environmental inputs, output actions of an agent, and corresponding pain or pleasure. The model network (e.g. (Werbos, 1987)(Jordan, 1988)(Robinson and Fallside, 1989)) in turn allows the system to compute controller gradients for 'minimizing pain' and 'maximizing pleasure'. *Since reinforcement gradients depend on 'credit assignment paths' leading 'backwards through the environment', the model network should not only predict the pain and pleasure units but also the other input units.*

The quantity to be minimized by the model network is $\sum_{t,i}(y_i(t) - y_{ipred}(t))^2$, where $y_i(t)$ is the activation of the $i$th input unit at time $t$, and $y_{ipred}(t)$ is the model's prediction of the activation of the $i$th input unit at time $t$. The quantity to be minimized by the controller is $\sum_{t,i}(c_i - r_i(t))^2$, where $r_i(t)$ is the activation of the $i$th pain or pleasure input unit at time $t$ and $c_i$ is its desired activation for all times. $t$ ranges over all (discrete) time steps. Weights are changed at each time step. This relieves dependence on 'episode boundaries'. Here the assumption is that the learning rates are small enough to avoid instabilities (Williams and Zipser, 1989).

There are two versions of the algorithm: the sequential version and the parallel version. With the sequential version, the model network is first trained by providing it with randomly chosen examples of sequences of interactions between controller and environment. Then the model's weights are fixed to their current values, and the controller begins to learn. With the parallel version both the controller and the model learn concurrently. One advantage of the parallel version is that the model network focusses only on those parts of the environmental dynamics with which the controller typically is confronted. Another advantage is the applicability to changing environments. Some disadvantages of the parallel version are listed next.

*1. Imperfect model networks.* The model which is used to compute gradient information for the controller may be wrong. However, *if we assume that the model network always finds a zero-point of its error function*, then over time we can expect the control network to perform gradient descent according to a perfect model of the visible parts of the real world. *1.A:* The assumption that the model network can always find a zero-point of its error function is not valid in the general case. One of the reasons is the old problem of local minima, for which this paper does not suggest any solutions. *1.B:* (Jordan, 1988) notes that a model network does not need to be perfect to allow increasing performance of the control network.

*2. Instabilities.* One source of instability could arise if the model network 'forgets' information about the environmental dynamics because the activities of the controller push it into a new sub-domain, such that the weights responsible for the old well-modeled sub-domain become over-written.

*3. Deadlock.* Even if the model's predictions are perfect for all actions executed by the controller, this does not imply that the algorithm will always behave as desired. Let us assume that the controller enters a local minimum relative to the current state of an imperfect model network. This relative minimum might cause the controller to execute the same action again and again (in a certain spatio-temporal context), while the model does not get a chance to learn something about the consequences of *alternative* actions (this is the *deadlock*).

The sequential version lacks the flavor of on-line learning and is bound to fail as soon as the environment changes significantly. We will introduce 'adaptive randomness' for the controller outputs to attack problems of the parallel version.

## 2    THE ALGORITHM

The sequential version of the algorithm can be obtained in a straight-forward manner from the description of the parallel version below. At every time step, the parallel version is performing essentially the same operations:

In step 1 of the main loop of the algorithm, actions to be performed in the external world are computed. These actions are based on both current and previous inputs and outputs. For all new activations, the corresponding derivatives with respect to all controller weights are updated. In step 2 actions are executed in the external world, and the effects of the current action and/or previous actions may become visible. In step 3 the model network sees the last input and the current output of the controller at the same time. The model network tries to predict the new input without seeing it. Again the relevant gradient information is computed. In step 4 the model network is updated in order to better predict the input (including pleasure and pain) for the controller. The weights of the control network are updated in order to minimize the cumulative differences between desired and actual activations of the pain and pleasure units. 'Teacher forcing' (Williams and Zipser, 1989) is used in the model network (although there is no teacher besides the environment). The partial derivatives of the controller's inputs with respect to the controller's weights are approximated by the partial derivatives of the corresponding predictions generated by the model network.

Notation (the reader may find it convenient to compare with (Williams and Zipser, 1989)): $C$ *is the set of all non-input units of the control network,* $A$ *is the set of its output units,* $I$ *is the set of its 'normal' input units,* $P$ *is the set of its pain and pleasure units,* $M$ *is the set of all units of the model network,* $O$ *is the set of its output units,* $O_P \subset O$ *is the set of all units that predict pain or pleasure,* $W_M$ *is the set of variables for the weights of the model network,* $W_C$ *is the set of variables for the weights of the control network,* $y_{k_{new}}$ *is the variable for the updated activation of the kth unit from* $M \cup C \cup I \cup P$, $y_{k_{old}}$ *is the variable for the last value of* $y_{k_{new}}$, $w_{ij}$ *is the variable for the weight of the directed connection from unit j to unit i.* $\delta_{ik}$ *is the Kronecker-delta, which is 1 for* $i = k$ *and 0 otherwise,* $p_{ij_{new}}^k$ *is the variable which gives the current (approximated) value of* $\frac{\partial y_{k_{new}}}{\partial w_{ij}}$, $p_{ij_{old}}^k$ *is the variable which gives the last value of* $p_{ij_{new}}^k$. *If* $k \in P$ *then* $c_k$ *is k's desired activation for all times, if* $k \in I \cup P$, *then kpred is the unit from* $O$ *which predicts k.* $\alpha_C$ *is the learning rate for the control network,* $\alpha_M$ *is the learning rate for the model network.*

$| I \cup P | = | O |$, $| O_P | = | P |$. *Each unit in* $I \cup P \cup A$ *has one forward connection to each unit in* $M \cup C$, *each unit in* $M$ *is connected to each other unit in* $M$, *each unit in* $C$ *is connected to each other unit in* $C$. *Each weight variable of a connection leading to a unit in* $M$ *is said to belong to* $W_M$, *each weight variable of a connection leading to a unit in* $C$ *is said to belong to* $W_C$. *For each weight* $w_{ij} \in W_M$ *there are* $p_{ij}^k$-*values for all* $k \in M$, *for each weight* $w_{ij} \in W_C$ *there are* $p_{ij}^k$-*values for all* $k \in M \cup C \cup I \cup P$. *The parallel version of the algorithm works as follows:*

*INITIALIZATION:*

$\forall\ w_{ij} \in W_M \cup W_C$: $w_{ij} \leftarrow random$, $\forall\ possible\ k$: $p^k_{ij_{old}} \leftarrow 0, p^k_{ij_{new}} \leftarrow 0$ .

$\forall\ k \in M \cup C$ : $y_{k_{old}} \leftarrow 0, y_{k_{new}} \leftarrow 0.$

$\forall\ k \in I \cup P$ : Set $y_{k_{old}}$ according to the current environment, $y_{k_{new}} \leftarrow 0.$

*UNTIL TERMINATION CRITERION IS REACHED :*

1. $\forall\ i \in C : y_{i_{new}} \leftarrow \dfrac{1}{1+e^{-\sum_j w_{ij}v_{j_{old}}}}.$

    $\forall\ w_{ij} \in W_C, k \in C: p^k_{ij_{new}} \leftarrow y_{k_{new}}(1 - y_{k_{new}})(\sum_l w_{kl}p^l_{ij_{old}} + \delta_{ik}y_{j_{old}}).$

    $\forall\ k \in C: y_{k_{old}} \leftarrow y_{k_{new}}, \forall\ w_{ij} \in W_C : p^k_{ij_{old}} \leftarrow p^k_{ij_{new}}$ .

2. *Execute all actions based on activations of units in A. Update the environment.*

    $\forall\ i \in I \cup P:$ Set $y_{i_{new}}$ according to environment.

3. $\forall\ i \in M : y_{i_{new}} \leftarrow \dfrac{1}{1+e^{-\sum_j w_{ij}v_{j_{old}}}}.$

    $\forall\ w_{ij} \in W_M \cup W_C, k \in M: p^k_{ij_{new}} \leftarrow y_{k_{new}}(1 - y_{k_{new}})(\sum_l w_{kl}p^l_{ij_{old}} + \delta_{ik}y_{j_{old}}).$

    $\forall\ k \in M: y_{k_{old}} \leftarrow y_{k_{new}}, \forall\ w_{ij} \in W_C \cup W_M : p^k_{ij_{old}} \leftarrow p^k_{ij_{new}}$ .

4. $\forall\ w_{ij} \in W_M: w_{ij} \leftarrow w_{ij} + \alpha_M \sum_{k \in I \cup P}(y_{k_{new}} - y_{kpred_{old}})p^{kpred}_{ij_{old}}$ .

    $\forall\ w_{ij} \in W_C: w_{ij} \leftarrow w_{ij} + \alpha_C \sum_{k \in P}(c_k - y_{k_{new}})p^{kpred}_{ij_{old}}.$

    $\forall\ k \in I \cup P: y_{k_{old}} \leftarrow y_{k_{new}}, y_{kpred_{old}} \leftarrow y_{k_{new}}, \forall\ w_{ij} \in W_M : p^{kpred}_{ij_{old}} \leftarrow 0,$

    $\forall\ w_{ij} \in W_C : p^k_{ij_{old}} \leftarrow p^{kpred}_{ij_{old}}$ .

The algorithm is local in time, but not in space. The computation complexity per time step is $O(|\ W_M \cup W_C\ ||\ M\ ||\ M \cup I \cup P \cup A\ | + |\ W_C\ ||\ C\ ||\ I \cup P \cup C\ |)$. In what follows we describe some useful extensions of the scheme.

*1. More network ticks than environmental ticks.* For highly 'non-linear' environments the algorithm has to be modified in a trivial manner such that the involved networks perform more than one (but not more than three) iterations of step 1 and step 3 at each time step. (4-layer-operations in principle can produce an arbitrary approximation of any desired mapping.)

*2. Adaptive randomness.* Explicit explorative random search capabilities can be introduced by probabilistic controller outputs and 'gradient descent through random number generators' (Williams, 1988). We adjust both the mean and the variance of the controller actions. In the context of the IID algorithm, this works as follows: A probabilistic output unit $k$ consists of a conventional unit $k\mu$ which acts as a mean generator and a conventional unit $k\sigma$ which acts as a variance generator. At a given time, the probabilistic output $y_{k_{new}}$ is computed by $y_{k_{new}} = y_{k\mu_{new}} + zy_{k\sigma_{new}}$, where $z$ is distributed e.g. according to the normal distribution. The corresponding $p^k_{ij_{new}}$

must then be updated according to the following rule:

$$p_{ij_{new}}^{k} \leftarrow p_{ij_{new}}^{k\mu} + \frac{y_{k_{new}} - y_{k\mu_{new}}}{y_{k\sigma_{new}}} p_{ij_{new}}^{k\sigma}.$$

A more sophisticated strategy to improve the model network is to introduce 'adaptive curiosity and boredom'. The priniciple of adaptive curiosity for model-building neural controllers (Schmidhuber, 1990a) says: Spend additional reinforcement whenever there is a mismatch between the expectations of the model network and reality.

*3. Perfect models.* Sometimes one can gain a 'perfect' model by constructing an appropriate mathematical description of the environmental dynamics. This saves the time needed to train the model. However, additional external knowledge is required. For instance, the description of the environment might be in form of differential or difference equations. In the context of the algorithm above, this means introducing new $p_{ij}^{\eta}$ variables for each $w_{ij} \in W_C$ and each relevant state variable $\eta(t)$ of the dynamical environment. The new variables serve to accumulate the values of $\frac{\partial \eta(t)}{\partial w_{ij}}$. This can be done in exactly the same cumulative manner as with the activations of the model network above.

*4. Augmenting the algorithm by TD-methods.* The following ideas are not limited to recurrent nets, but are also relevant for feed-forward controllers in Markovian environments.

It is possible to augment model-building algorithms with an 'adaptive critic' method. To simplify the discussion, let us assume that there are no pleasure units, just pain units. The algorithm's goal is to minimize cumulative pain. We introduce the TD-principle (Sutton, 1988) by changing the error function of the units in $O_P$: At a given time $t$, the contribution of each unit $kpred \in O_P$ to the model network's error is $y_{kpred}(t) - \gamma y_{kpred}(t+1) - y_k(t+1)$, where $y_i(t)$ is the activation of unit $i$ at time $t$, and $0 < \gamma < 1$ is a discount factor for avoiding predictions of infinite sums. Thus $O_P$ is trained to predict the *sum of all* (discounted) future pain vectors and becomes a *vector-valued adaptive critic*. (This affects the first ∀-*loop* in step 4 .)

The controller's goal is to minimize the absolute value of $M$'s pain predictions. Thus, the contribution of time $t$ to the error function of the controller now becomes $\sum_{kpred \in O_P} (y_{kpred}(t))^2$. This affects the second *For-loop* in step 4 of the algorithm. Note that it is not a *state* which is evaluated by the adaptive critic component, but a *combination of a state and an action*. This makes the approach similar to (Jordan and Jacobs, 1990). (Schmidhuber, 1990a) shows how a recurrent model/controller combination can be used for look-ahead planning *without* using TD-methods.

## 3   EXPERIMENTS

The following experiments were conducted by the TUM-students Josef Hochreiter and Klaus Bergner. See (Schmidhuber, 1990a) and (Schmidhuber, 1990b) for the full details.

*1. Evolution of a flip-flop by reinforcement learning.* A controller $K$ had to learn to behave like a flip-flop as described in (Williams and Zipser, 1989). The *main diffi-*

*culty* (the one which makes this different from the supervised approach as described in (Williams and Zipser, 1989)) was that there was no teacher for $K$'s (probabilistic) output units. Instead, the system had to generate alternative outputs in a variety of spatio-temporal contexts, and to build a model of the often 'painful' consequences. $K$'s only goal information was the activation of a pain input unit whenever it produced an incorrect output. With $| C |= 3$, $| M |= 4$, $\alpha_C = 0.1$ and $\alpha_M = 1.0$ 20 out of 30 test runs with the parallel version required less than 1000000 time steps to produce an acceptable solution.

Why does it take much more time solving the reinforcement flip-flop problem than solving the corresponding supervised flip-flop problem? One answer is: With supervised learning the controller gradient is *given* to the system, while with reinforcement learning the gradient has to be *discovered* by the system.

*2. 'Non-Markovian' pole balancing.* A cart pole system was modeled by the same differential equations used for a related balancing task which is described in (Anderson, 1986). In contrast to previous pole balancing tasks, however, *no information about temporal derivatives of cart position and pole angle was provided.* (Similar experiments are mentioned in (Piché, 1990).)

In our experiments the cart-pole system would not stabilize indefinitely. However, significant performance improvement was obtained. The best results were achieved by using a 'perfect model' as described above: Before learning, the average time until failure was about 25 time steps. Within a few hundred trials one could observe trials with more than 1000 time steps balancing time. 'Friendly' initial conditions could lead to balancing times of more than 3000 time steps.

*3. 'Markovian' pole balancing with a vector-valued adaptive critic.* The adaptive critic extension described above does not need a non-Markovian environment to demonstrate advantages over previous adaptive critics: A four-dimensional adaptive critic was tested on the pole balancing task described in (Anderson, 1986). The critic component had four output units for predicting four different kinds of 'pain', two for bumps against the two edges of the track and two for pole crashes.

None of five conducted test runs took more than 750 failures to achieve the first trial with more than 30000 time steps. (The longest run reported by (Anderson, 1986) took about 29000 time steps, more than 7000 failures had to be experienced to achieve that result.)

# 4   SOME LIMITATIONS OF THE APPROACHES

1. The recurrent network algorithms are not local in space.

2. As with all gradient descent algorithms there is the problem of local minima. This paper does not offer any solutions to this problem.

3. More severe limitations of the algorithm are *inherent problems* of the concepts of 'gradient descent through time' and adaptive critics. Neither gradient descent nor adaptive critics are practical when there are *long* time lags between actions and ultimate consequences. For this reason, first steps are made in (Schmidhuber, 1990c) towards *adaptive sub-goal generators* and *adaptive 'causality detectors'*.

**Acknowledgements**

I wish to thank Josef Hochreiter and Klaus Bergner who conducted the experiments. This work was supported by a scholarship from SIEMENS AG.

# References

Anderson, C. W. (1986). *Learning and Problem Solving with Multilayer Connectionist Systems*. PhD thesis, University of Massachusetts, Dept. of Comp. and Inf. Sci.

Jordan, M. I. (1988). Supervised learning and systems with excess degrees of freedom. Technical Report COINS TR 88-27, MIT.

Jordan, M. I. and Jacobs, R. A. (1990). Learning to control an unstable system with forward modeling. In *Proc. of the 1990 Connectionist Models Summer School, in press.* San Mateo, CA: Morgan Kaufmann.

Piché, S. W. (1990). Draft: First order gradient descent training of adaptive discrete time dynamic networks. Technical report, Dept. of Electrical Engineering, Stanford University.

Robinson, A. J. and Fallside, F. (1987). The utility driven dynamic error propagation network. Technical Report CUED/F-INFENG/TR.1, Cambridge University Engineering Department.

Robinson, T. and Fallside, F. (1989). Dynamic reinforcement driven error propagation networks with application to game playing. In *Proceedings of the 11th Conference of the Cognitive Science Society, Ann Arbor*, pages 836–843.

Schmidhuber, J. H. (1990a). Making the world differentiable: On using fully recurrent self-supervised neural networks for dynamic reinforcement learning and planning in non-stationary environments. Technical Report FKI-126-90 (revised), Institut für Informatik, Technische Universität München. (Revised and extended version of an earlier report from February.).

Schmidhuber, J. H. (1990b). Networks adjusting networks. Technical Report FKI-125-90 (revised), Institut für Informatik, Technische Universität München. (Revised and extended version of an earlier report from February.).

Schmidhuber, J. H. (1990c). Towards compositional learning with dynamic neural networks. Technical Report FKI-129-90, Institut für Informatik, Technische Universität München.

Sutton, R. S. (1988). Learning to predict by the methods of temporal differences. *Machine Learning*, 3:9–44.

Werbos, P. J. (1987). Building and understanding adaptive systems: A statistical/numerical approach to factory automation and brain research. *IEEE Transactions on Systems, Man, and Cybernetics*, 17.

Williams, R. J. (1988). On the use of backpropagation in associative reinforcement learning. In *IEEE International Conference on Neural Networks, San Diego*, volume 2, pages 263–270.

Williams, R. J. and Zipser, D. (1989). Experimental analysis of the real-time recurrent learning algorithm. *Connection Science*, 1(1):87–111.